# Order Reduction for Dynamical Systems Describing the Behavior of Complex Neurons

Thomas B. Kepler          L. F. Abbott          Eve Marder
Biology Dept.             Physics Dept.          Biology Dept.

Brandeis University
Waltham, MA 02254

## Abstract

We have devised a scheme to reduce the complexity of dynamical systems belonging to a class that includes most biophysically realistic neural models. The reduction is based on transformations of variables and perturbation expansions and it preserves a high level of fidelity to the original system. The techniques are illustrated by reductions of the Hodgkin-Huxley system and an augmented Hodgkin-Huxley system.

## INTRODUCTION

For almost forty years, biophysically realistic modeling of neural systems has followed the path laid out by Hodgkin and Huxley (Hodgkin and Huxley, 1952). Their seminal work culminated in the accurately detailed description of the membrane currents expressed by the giant axon of the squid *Loligo*, as a system of four coupled non-linear differential equations. Soon afterward (and ongoing now) simplified, abstract models were introduced that facilitated the conceptualization of the model's behavior, *e.g.* (FitzHugh, 1961). Yet the mathematical relationships between these conceptual models and the realistic models have not been fully investigated. Now that neurophysiology is telling us that most neurons are complicated and subtle dynamical systems, this situation is in need of change. We suggest that a systematic program of simplification in which a realistic model of given complexity spawns a family of simplified meta-models of varying degrees of abstraction could yield considerable advantage. In any such scheme, the number of dynamical variables, or order, must be reduced, and it seems efficient and reasonable to do this first. This paper will be concerned with this step only. A sketch of a more thoroughgoing scheme proceeding ultimately to the binary formal neurons of

Hopfield (Hopfield, 1982) has been presented elsewhere (Abbott and Kepler, 1990). There are at present several reductions of the Hodgkin-Huxley (HH) system (FitzHugh, 1961; Krinskii and Kokoz, 1973; Rose and Hindmarsh, 1989) but all of them suffer to varying degrees from a lack of generality and/or insufficient realism.

We will present a scheme of perturbation analyses which provide a power-series approximation of the original high-order system and whose leading term is a lower-order system (see (Kepler et al., 1991) for a full discussion). The techniques are general and can be applied to many models. Along the way we will refer to the HH system for concreteness and illustrations. Then, to demonstrate the generality of the techniques and to exhibit the theoretical utility of our approach, we will incorporate the transient outward current described in (Connor and Stevens, 1972) and modeled in (Connor et al., 1977) known as $I_A$. We will reduce the resulting sixth-order system to both third- and second-order systems.

## EQUIVALENT POTENTIALS

Many systems modeling excitable neural membrane consist of a differential equation expressing current conservation

$$C\frac{dV}{dt} + I(V,\{x_i\}) = I_{external}(t) \qquad (1)$$

where V is the membrane potential difference, C is the membrane capacitance and I(V,x) is the total ionic current expressed as a function of V and the $x_i$, which are gating variables described by equations of the form

$$\frac{dx_i}{dt} = k_i(V)(\bar{x}_i(V) - x_i). \qquad (2)$$

providing the balance of the system's description. The ubiquity of the membrane potential and its role as "command variable" in these model systems suggests that we might profit by introducing potential-like variables in place of the gating variables. We define the equivalent potential (EP) for each of the gating variables by

$$V_i = \bar{x}_i^{-1}(x_i). \qquad (3)$$

In realistic neural models, the function $\bar{x}$ is ordinarily sigmoid and hence invertible. The chain rule may be applied to give us new equations of motion. Since no approximations have yet been made, the system expressed in these variables gives exactly the same evolution for V as the original system. The evolution of the whole HH system expressed in EPs is shown in fig. 1. There is something striking about this collection of plots. The transformation to EPs now suggests that of the four available degrees of freedom, only two are actually utilized. Specifically, $V_m$ is nearly indistinguishable from V, and $V_h$ and $V_n$ are likewise quite similar. This strongly suggests that we form averages and differences of EPs within the two classes.

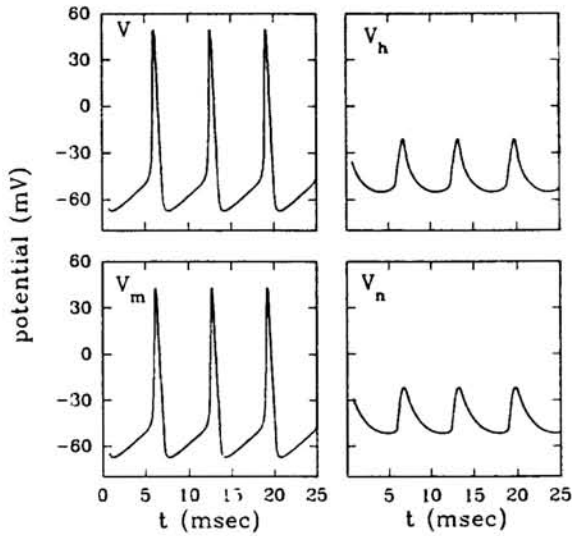

*Figure 1: Behavior of equivalent potentials in repetetive firing mode of Hodgkin-Huxley system.*

## PERTURBATION SERIES

In the general situation, the EPs must be segregated into two or more classes. One class will contain the true membrane potential V. Members of this class will be subscripted with greek letters $\mu$, $\nu$, etc. while the others will be subscripted with latin indices i, j, etc. We make, within each class, a change of variables to 1) a new *representative* EP taken as a weighted average over all members of the class, and 2) differences between each member and their average. The transformations and their inverses are

$$\phi = \sum_\mu \alpha_\mu V_\mu \qquad \psi = \sum_i \alpha_i V_i$$

$$\delta_\mu = V_\mu - <V_\nu> \qquad \delta_i = V_i - <V_j> \tag{4}$$

and

$$V_\mu = \phi - \sum_\nu \alpha_\nu \delta_\nu + \delta_\mu \qquad V_i = \psi - \sum_j \alpha_j \delta_j + \delta_i. \tag{5}$$

We constrain the $\alpha_i$ and the $\alpha_\mu$ to sum to one. The $\alpha$'s will not be taken as constants, but will be allowed to depend on $\phi$ and $\psi$. We expect, however, that their variation will be small so that most of the time dependence of $\phi$ and $\psi$ will be carried by the V's. We differentiate eqs.(4), use the inverse transformations of eq.(5) and expand to first order in the $\delta$'s to get

$$\frac{d\psi}{dt} = \sum_i \alpha_i \, k_i(\phi)\left(\bar{x}_i(\phi) - \bar{x}_i(\psi)\right) + O(\delta) \tag{6}$$

and the new current conservation equation,

$$C\frac{d\phi}{dt} + \alpha_0 I(\phi, \{\bar{x}_\mu(\phi)\}, \{\bar{x}_i(\psi)\}) = I_{external}(t) + O(\delta). \tag{7}$$

This is still a current conservation equation, only now we have renormalized the capacitance in a state-dependent way through $\alpha_0$. The coefficient the of $\delta$'s in eq.(6) will be small, at least in the neighborhood of the equilibrium point, as long as the basic premise of the expansion holds. No such guarantee is made about the corresponding

coefficient in eq.(7). Therefore we will choose the $\alpha$'s to make the correction term *second* order in the $\delta$'s by setting the coefficient of each $\delta_i$ and $\delta_\mu$ to zero. For the $\delta_\mu$ we get,

$$\alpha_0 \alpha_\mu A - \alpha_0 I_{,\mu} + C\alpha_\mu k_\mu + C\dot{\alpha}_\mu = 0 \tag{8}$$

for $\mu \neq 0$, where

$$I_{,j} \equiv \frac{\partial I}{\partial x_j} \vec{x}_j' \tag{9}$$

and we use the abbreviation $A \equiv \Sigma\, I_{,\mu}$. And for $\mu = 0$,

$$\alpha_0^2 A - \alpha_0 I_{,0} - C\sum_{\nu \neq 0} \alpha_\nu k_\nu + C\dot{\alpha}_0 = 0 \tag{10}$$

Now the time derivatives of the $\alpha$'s vanish at the equilibrium point, and it is with the neighborhood of this point that we must be primarily concerned. Ignoring these terms yields surprisingly good results even far from equilibrium. This choice having been made, we solve for $\alpha_0$, as the root of the polynomial

$$\alpha_0 A - I_{,0} - C\sum_{\mu \neq 0} k_\mu I_{,\mu} [\alpha_0 A + C k_\mu]^{-1} = 0 \tag{11}$$

whose order is equal to the number of EPs combining to form $\phi$. The time dependence of $\psi$ is given by specifying the $\alpha_i$. This may be done as for the $\alpha_\mu$ to get

$$\alpha_i = I_{,i} \left[\sum_j I_{,j}\right]^{-1} \tag{12}$$

## EXAMPLE: HODGKIN-HUXLEY + $I_A$

For the specific cases in which the HH system is reduced from fourth order to second, by combining V and $V_m$ to form $\phi$ and combining $V_h$ and $V_n$ to form $\psi$, the plan outlined above works without any further meddling, and yields a very faithful reduction. Also straightforward is the reduction of the sixth-order system given by Connor et al. (Connor et al., 1977) in which the HH system is supplemented by $I_A$ (HH+A) to third order. In this reduction, the EP for the $I_A$ activation variable, a, joins $V_h$ and $V_n$ in the formation of $\psi$. Alternatively, we may reduce to a second order system in which $V_a$ joins with V and $V_m$ to form $\phi$ and the EPs for n,h and the $I_A$ inactivation variable, b, are combined to form $\psi$. This is not as straightforward. A direct application of eq.(12) produces a curve of singularities where the denominator vanishes in the expression for $d\psi/dt$; on one side $d\psi/dt$ has the same sign as $\phi - \psi$, (which it should) and on the other side it does not. Some additional decisions must be made here. We may certainly take this to be an indication that the reduction is breaking down, but through good fortune we are able to salvage it. This matter is dealt with in more detail elsewhere (Kepler et al., 1991). The reduced models are related in that the first is recovered when the maximum conductance

of $I_A$ is set to zero in either of the other two.

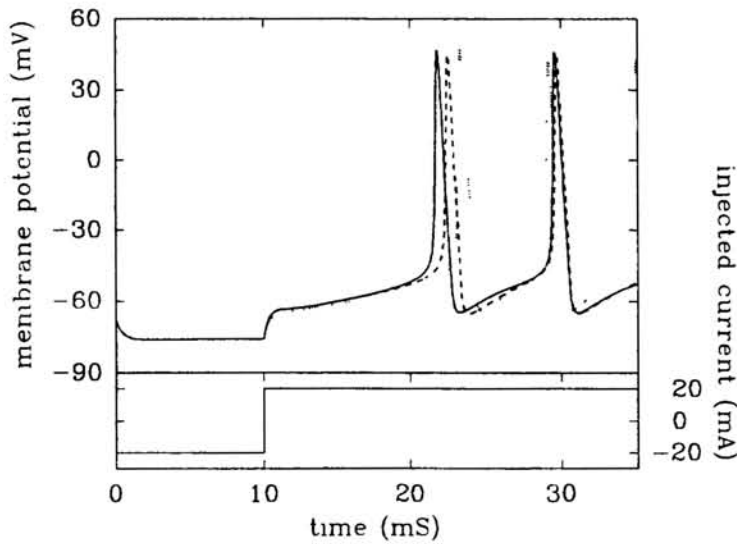

Figure 2: *Response of full HH+A (solid line), 3ʳᵈ order (dashed) and 2ⁿᵈ order systems to current step, showing latency to firing.*

Figure 2 shows the voltage trace of a HH+A cell that is first hyperpolarized and then suddenly depolarized to above threshold. Traces from all three systems (full, 3ʳᵈ order, 2ⁿᵈ order) are shown superimposed. This example focuses on the phenomenon of post inhibitory latency to firing. When a HH cell is depolarized sufficiently to produce firing, the onset of the first action potential is immediate and virtually independent of the degree of hyperpolarization experienced immediately beforehand. In contrast, the same cell with an $I_A$ now shows a latency to firing which depends monotonically on the depth to which it had been hyperpolarized immediately prior to depolarization.

This is most clearly seen in fig.3 showing the phase portrait of the second-order system. The $d\phi/dt = 0$ nullcline has acquired a second branch. In order to get from the initial (hyperpolarized) location, the phase point must crawl over this obstacle, and the lower it starts, the farther it has to climb.

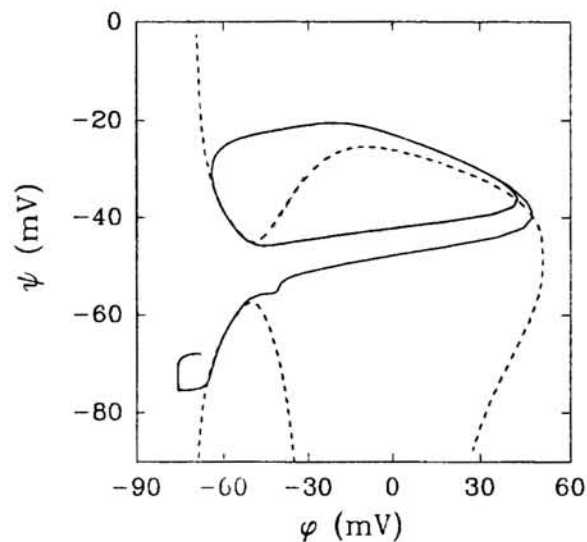

Figure 3: *Phase portrait of event shown in fig.3 for 2ⁿᵈ order reduced system.*

Figure 4 shows the firing frequency as a function of the injected current, for the full HH and HH+A sytems (solid lines), the HH second order HH+A third order (dashed lines) and HH+A second order

(dotted line).* Note that the first reduction matches the full system quite well in both cases. The second reduction, however, does not do as well. It does get the qualitative features right, though. The expansion of the dynamic range for the frequency of firing is still present, though squeezed into a much smaller interval on the current axis. The bifurcation occurs at nearly the right place and seems to have the proper character, *i.e.*, saddle rather than Hopf, though this has not been rigorously investigated.

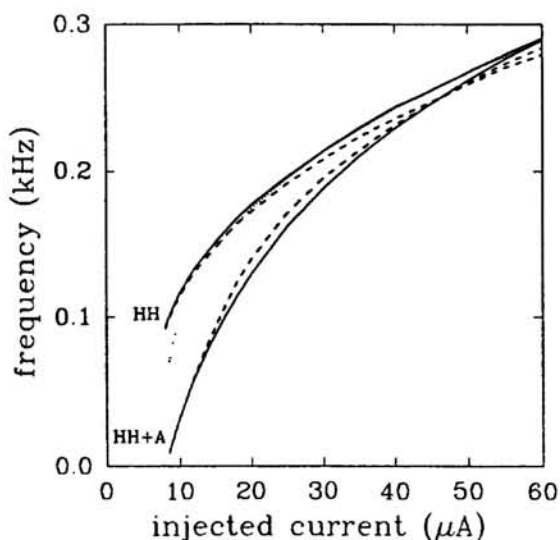

*Figure 4*: *Firing frequency as a function of injected current. Solid: full systems, dashed: $2^{nd}$ order HH & $3^{rd}$ order HH+A, dotted: $2^{nd}$ order HH+A. (From Kepler et al., 1991)*

The reduced systems are intended to be dynamically realistic, to respond accurately to the kind of time-dependent external currents that would be encountered in real networks. To put this to the test, we ran simulations in which $I_{external}(t)$ was given by a sum of sinusoids of equal amplitude and randomly chosen frequency and phase. Figure 5 illustrates the remarkable match between the full HH+A system and the third-order reduction, when such an irregular (quasiperiodic) current signal is used to drive them.

## CONCLUSION

We have presented a systematic approach to the reduction of order for a class of dynamical systems that includes the Hodgkin-Huxley system, the Connor et al. $I_A$ extension of the HH system, and many other realistic neuron models. As mentioned at the outset, these procedures are merely the first steps in a more comprehensive program of simplification. In this way, the conceptual advantage of abstract models may be joined to the biophysical realism of physiologically derived models in a smooth and tractable manner, and the benefits of simplicity may be enjoyed with a clear conscience.

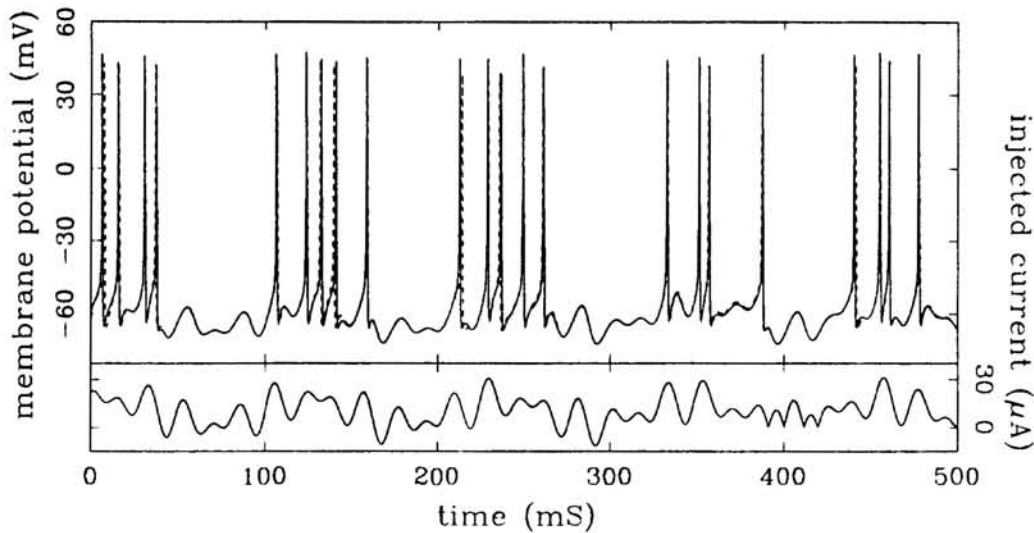

*Figure 5*: *Response of HH+A system to irregular current injection. Solid line: full system, dashed line: 3^rd order reduction.*

## Acknowledgment

This work was supported by National Institutes of Health grant T32NS07292 (TBK), Department of Energy Contract DE-AC0276-ER03230 (LFA) and National Institutes of Mental Health grant MH46742 (EM).

## Footnotes

*For purposees of comparison, the HH system used here is as modified by (Connor et al., 1977), but with $I_A$ removed and the leakage reversal potential adjusted to give the same resting potential as the HH+A cell.

## REFERENCES

L.F.Abbott and T.B.Kepler, 1990 in *Proceedings of the XI Sitges Conference on Neural Networks* in press

J.A.Connor and C.F.Stevens, 1971 *J.Physiol.,Lond.* **213** 31

J.A.Connor, D.Walter and R.McKown, 1977 *Biophys.J.* **18** 81

R.FitzHugh, 1961 *Biophys. J.* **1** 445

A.L.Hodgkin and A.F.Huxley, 1952 *J. Physiol.* **117**, 500

J.J.Hopfield, 1982 *Proc.Nat.Acad.Sci.* **79** 2554

T.B.Kepler, L.F. Abbott and E.Marder, 1991 submitted to *Biol.Cyber.*

V.I.Krinskii and Yu.M.Kokoz, 1973 *Biofizika* **18** 506

R.M.Rose and J.L.Hindmarsh, 1989 *Proc.R.Soc.Lond.* **237** 267